# The Curse of Highly Variable Functions for Local Kernel Machines

**Yoshua Bengio, Olivier Delalleau, Nicolas Le Roux**
Dept. IRO, Université de Montréal
P.O. Box 6128, Downtown Branch, Montreal, H3C 3J7, Qc, Canada
{*bengioy,delallea,lerouxni*}*@iro.umontreal.ca*

## Abstract

We present a series of theoretical arguments supporting the claim that a large class of modern learning algorithms that rely solely on the smoothness prior – with similarity between examples expressed with a local kernel – are sensitive to the curse of dimensionality, or more precisely to the variability of the target. Our discussion covers supervised, semi-supervised and unsupervised learning algorithms. These algorithms are found to be local in the sense that crucial properties of the learned function at $x$ depend mostly on the neighbors of $x$ in the training set. This makes them sensitive to the curse of dimensionality, well studied for classical non-parametric statistical learning. We show in the case of the Gaussian kernel that when the function to be learned has many variations, these algorithms require a number of training examples proportional to the number of variations, which could be large even though there may exist short descriptions of the target function, i.e. their Kolmogorov complexity may be low. This suggests that there exist non-local learning algorithms that at least have the potential to learn about such structured but apparently complex functions (because locally they have many variations), while not using very specific prior domain knowledge.

## 1 Introduction

A very large fraction of the recent work in statistical machine learning has been focused on non-parametric learning algorithms which rely solely, explicitly or implicitly, on the **smoothness prior**, which says that we prefer as solution functions $f$ such that when $x \approx y$, $f(x) \approx f(y)$. Additional prior knowledge is expressed by choosing the space of the data and the particular notion of similarity between examples (typically expressed as a kernel function). This class of learning algorithms therefore includes most of the kernel machine algorithms (Schölkopf, Burges and Smola, 1999), such as Support Vector Machines (SVMs) (Boser, Guyon and Vapnik, 1992; Cortes and Vapnik, 1995) or Gaussian processes (Williams and Rasmussen, 1996), but also unsupervised learning algorithms that attempt to capture the manifold structure of the data, such as Locally Linear Embedding (Roweis and Saul, 2000), Isomap (Tenenbaum, de Silva and Langford, 2000), kernel PCA (Schölkopf, Smola and Müller, 1998), Laplacian Eigenmaps (Belkin and Niyogi, 2003), Manifold Charting (Brand, 2003), and *spectral clustering* algorithms (see (Weiss, 1999) for a review). More recently, there has also been much interest in non-parametric *semi-supervised learning algorithms*, such as (Zhu, Ghahramani and Lafferty, 2003; Zhou

et al., 2004; Belkin, Matveeva and Niyogi, 2004; Delalleau, Bengio and Le Roux, 2005), which also fall in this category, and share many ideas with manifold learning algorithms.

Since this is a very large class of algorithms and it is attracting so much attention, it is worthwhile to investigate its limitations, and this is the main goal of this paper. Since these methods share many characteristics with classical non-parametric statistical learning algorithms (such as the $k$-nearest neighbors and the Parzen windows regression and density estimation algorithms (Duda and Hart, 1973)), which have been shown to suffer from the so-called *curse of dimensionality*, it is logical to investigate the following question: to what extent do these modern kernel methods suffer from a similar problem?

In this paper, we focus on algorithms in which the learned function is expressed in terms of a linear combination of kernel functions applied on the training examples:

$$f(x) = b + \sum_{i=1}^{n} \alpha_i K_D(x, x_i) \tag{1}$$

where optionally a bias term $b$ is added, $D = \{z_1, \ldots, z_n\}$ are training examples ($z_i = x_i$ for unsupervised learning, $z_i = (x_i, y_i)$ for supervised learning, and $y_i$ can take a special "missing" value for semi-supervised learning). The $\alpha_i$'s are scalars chosen by the learning algorithm using $D$, and $K_D(\cdot, \cdot)$ is the kernel function, a symmetric function (sometimes expected to be positive definite), which may be chosen by taking into account all the $x_i$'s. A typical kernel function is the Gaussian kernel,

$$K_\sigma(u, v) = e^{-\frac{1}{\sigma^2} ||u-v||^2}, \tag{2}$$

with the width $\sigma$ controlling how local the kernel is. See (Bengio et al., 2004) to see that LLE, Isomap, Laplacian eigenmaps and other spectral manifold learning algorithms such as spectral clustering can be generalized to be written as in eq. 1 for a test point $x$.

One obtains consistency of classical non-parametric estimators by appropriately varying the hyper-parameter that controls the locality of the estimator as $n$ increases. Basically, the kernel should be allowed to become more and more local, so that statistical bias goes to zero, but the "effective number of examples" involved in the estimator at $x$ (equal to $k$ for the $k$-nearest neighbor estimator) should increase as $n$ increases, so that statistical variance is also driven to 0. For a wide class of kernel regression estimators, the unconditional variance and squared bias can be shown to be written as follows (Härdle et al., 2004):

$$\text{expected error} = \frac{C_1}{n\sigma^d} + C_2\sigma^4,$$

with $C_1$ and $C_2$ not depending on $n$ nor on the dimension $d$. Hence an optimal bandwidth is chosen proportional to $n^{\frac{-1}{4+d}}$, and the resulting generalization error (not counting the noise) converges in $n^{-4/(4+d)}$, which becomes very slow for large $d$. Consider for example the increase in number of examples required to get the same level of error, in 1 dimension versus $d$ dimensions. If $n_1$ is the number of examples required to get a level of error $e$, to get the same level of error in $d$ dimensions requires on the order of $n_1^{(4+d)/5}$ examples, i.e. the **required number of examples is exponential in** $d$. For the $k$-nearest neighbor classifier, a similar result is obtained (Snapp and Venkatesh, 1998):

$$\text{expected error} = E_\infty + \sum_{j=2}^{\infty} c_j n^{-j/d}$$

where $E_\infty$ is the asymptotic error, $d$ is the dimension and $n$ the number of examples.

Note however that, if the data distribution is concentrated on a lower dimensional manifold, it is the **manifold dimension** that matters. Indeed, for data on a smooth lower-dimensional manifold, the only dimension that say a $k$-nearest neighbor classifier sees is the dimension

of the manifold, since it only uses the Euclidean distances between the near neighbors, and if they lie on such a manifold then the local Euclidean distances approach the local geodesic distances on the manifold (Tenenbaum, de Silva and Langford, 2000).

## 2 Minimum Number of Bases Required

In this section we present results showing the number of required bases (hence of training examples) of a kernel machine with Gaussian kernel may grow linearly with the "variations" of the target function that must be captured in order to achieve a given error level.

### 2.1 Result for Supervised Learning

The following theorem informs us about the number of sign changes that a Gaussian kernel machine can achieve, when it has $k$ bases (i.e. $k$ support vectors, or at least $k$ training examples).

**Theorem 2.1 (Theorem 2 of (Schmitt, 2002)).** *Let $f : \mathbb{R} \to \mathbb{R}$ computed by a Gaussian kernel machine (eq. 1) with $k$ bases (non-zero $\alpha_i$'s). Then $f$ has at most $2k$ zeros.*

We would like to say something about kernel machines in $\mathbb{R}^d$, and we can do this simply by considering a straight line in $\mathbb{R}^d$ and the number of sign changes that the solution function $f$ can achieve along that line.

**Corollary 2.2.** *Suppose that the learning problem is such that in order to achieve a given error level for samples from a distribution $P$ with a Gaussian kernel machine (eq. 1), then $f$ must change sign at least $2k$ times along some straight line (i.e., in the case of a classifier, the decision surface must be crossed at least $2k$ times by that straight line). Then the kernel machine must have at least $k$ bases (non-zero $\alpha_i$'s).*

*Proof.* Let the straight line be parameterized by $x(t) = u + tw$, with $t \in \mathbb{R}$ and $\|w\| = 1$ without loss of generality. Define $g : \mathbb{R} \to \mathbb{R}$ by

$$g(t) = f(u + tw).$$

If $f$ is a Gaussian kernel classifier with $k'$ bases, then $g$ can be written

$$g(t) = b + \sum_{i=1}^{k'} \beta_i \exp\left(-\frac{(t - t_i)^2}{2\sigma^2}\right)$$

where $u + t_i w$ is the projection of $x_i$ on the line $D_{u,w} = \{u + tw, t \in \mathbb{R}\}$, and $\beta_i \neq 0$. The number of bases of $g$ is $k'' \leq k'$, as there may exist $x_i \neq x_j$ such that $t_i = t_j$. Since $g$ must change sign at least $2k$ times, thanks to theorem 2.1, we can conclude that $g$ has at least $k$ bases, i.e. $k \leq k'' \leq k'$. $\qquad\square$

The above theorem tells us that if we are trying to represent a function that locally varies a lot (in the sense that its sign along a straight line changes many times), then we need many training examples to do so with a Gaussian kernel machine. Note that it says nothing about the dimensionality of the space, but we might expect to have to learn functions that vary more when the data is high-dimensional. The next theorem confirms this suspicion in the special case of the $d$-bits parity function:

$$\text{parity} : (b_1, \ldots, b_d) \in \{0, 1\}^d \mapsto \begin{cases} 1 \text{ if } \sum_{i=1}^d b_i \text{ is even} \\ -1 \text{ otherwise} \end{cases}$$

We will show that learning this apparently simple function with Gaussians centered on points in $\{0, 1\}^d$ is difficult, in the sense that it requires a number of Gaussians exponential in $d$ (for a fixed Gaussian width). Note that our corollary 2.2 does not apply to the $d$-bits

parity function, so it represents another type of local variation (not along a line). However, we are also able to prove a strong result about that case. We will use the following notations:

$$
\begin{aligned}
X_d &= \{0,1\}^d = \{x_1, x_2, \dots, x_{2^d}\} \\
H_d^0 &= \{(b_1, \dots, b_d) \in X_d \mid b_d = 0\} \\
H_d^1 &= \{(b_1, \dots, b_d) \in X_d \mid b_d = 1\}
\end{aligned}
\tag{3}
$$

$$\tag{4}$$

We say that a decision function $f : \mathbb{R}^d \to \mathbb{R}$ solves the parity problem if $\mathrm{sign}(f(x_i)) = \mathrm{parity}(x_i)$ for all $i$ in $\{1, \dots, 2^d\}$.

**Lemma 2.3.** *Let $f(x) = \sum_{i=1}^{2^d} \alpha_i K_\sigma(x_i, x)$ be a linear combination of Gaussians with same width $\sigma$ centered on points $x_i \in X_d$. If $f$ solves the parity problem, then $\alpha_i \mathrm{parity}(x_i) > 0$ for all $i$.*

*Proof.* We prove this lemma by induction on $d$. If $d = 1$ there are only 2 points. Obviously one Gaussian is not enough to classify correctly $x_1$ and $x_2$, so both $\alpha_1$ and $\alpha_2$ are non-zero, and $\alpha_1 \alpha_2 < 0$ (otherwise $f$ is of constant sign). Without loss of generality, assume $\mathrm{parity}(x_1) = 1$ and $\mathrm{parity}(x_2) = -1$. Then $f(x_1) > 0 > f(x_2)$, which implies $\alpha_1(1 - K_\sigma(x_1, x_2)) > \alpha_2(1 - K_\sigma(x_1, x_2))$ and $\alpha_1 > \alpha_2$ since $K_\sigma(x_1, x_2) < 1$. Thus $\alpha_1 > 0$ and $\alpha_2 < 0$, i.e. $\alpha_i \mathrm{parity}(x_i) > 0$ for $i \in \{1, 2\}$.

Suppose now lemma 2.3 is true for $d = d' - 1$, and consider the case $d = d'$. We denote by $x_i^0$ the points in $H_d^0$ and by $\alpha_i^0$ their coefficient in the expansion of $f$ (see eq. 3 for the definition of $H_d^0$). For $x_i^0 \in H_d^0$, we denote by $x_i^1 \in H_d^1$ its projection on $H_d^1$ (obtained by setting its last bit to 1), whose coefficient in $f$ is $\alpha_i^1$. For any $x \in H_d^0$ and $x_j^1 \in H_d^1$ we have:

$$
\begin{aligned}
K_\sigma(x_j^1, x) &= \exp\left(-\frac{\|x_j^1 - x\|^2}{2\sigma^2}\right) = \exp\left(-\frac{1}{2\sigma^2}\right) \exp\left(-\frac{\|x_j^0 - x\|^2}{2\sigma^2}\right) \\
&= \gamma K_\sigma(x_j^0, x)
\end{aligned}
$$

where $\gamma = \exp\left(-\frac{1}{2\sigma^2}\right) \in (0, 1)$. Thus $f(x)$ for $x \in H_d^0$ can be written

$$
\begin{aligned}
f(x) &= \sum_{x_i^0 \in H_d^0} \alpha_i^0 K_\sigma(x_i^0, x) + \sum_{x_j^1 \in H_d^1} \alpha_j^1 \gamma K_\sigma(x_j^0, x) \\
&= \sum_{x_i^0 \in H_d^0} \left(\alpha_i^0 + \gamma \alpha_i^1\right) K_\sigma(x_i^0, x).
\end{aligned}
$$

Since $H_d^0$ is isomorphic to $X_{d-1}$, the restriction of $f$ to $H_d^0$ implicitely defines a function over $X_{d-1}$ that solves the parity problem (because the last bit in $H_d^0$ is 0, the parity is not modified). Using our induction hypothesis, we have that for all $x_i^0 \in H_d^0$:

$$
\left(\alpha_i^0 + \gamma \alpha_i^1\right) \mathrm{parity}(x_i^0) > 0.
\tag{5}
$$

A similar reasoning can be made if we switch the roles of $H_d^0$ and $H_d^1$. One has to be careful that the parity is modified between $H_d^1$ and its mapping to $X_{d-1}$ (because the last bit in $H_d^1$ is 1). Thus we obtain that the restriction of $(-f)$ to $H_d^1$ defines a function over $X_{d-1}$ that solves the parity problem, and the induction hypothesis tells us that for all $x_j^1 \in H_d^1$:

$$
\left(-\left(\alpha_j^1 + \gamma \alpha_j^0\right)\right)\left(-\mathrm{parity}(x_j^1)\right) > 0.
\tag{6}
$$

and the two negative signs cancel out. Now consider any $x_i^0 \in H_d^0$ and its projection $x_i^1 \in H_d^1$. Without loss of generality, assume $\mathrm{parity}(x_i^0) = 1$ (and thus $\mathrm{parity}(x_i^1) = -1$). Using eq. 5 and 6 we obtain:

$$
\begin{aligned}
\alpha_i^0 + \gamma \alpha_i^1 &> 0 \\
\alpha_i^1 + \gamma \alpha_i^0 &< 0
\end{aligned}
$$

It is obvious that for these two equations to be simultaneously verified, we need $\alpha_i^0$ and $\alpha_i^1$ to be non-zero and of opposite sign. Moreover, because $\gamma \in (0,1)$, $\alpha_i^0 + \gamma\alpha_i^1 > 0 > \alpha_i^1 + \gamma\alpha_i^0 \Rightarrow \alpha_i^0 > \alpha_i^1$, which implies $\alpha_i^0 > 0$ and $\alpha_i^1 < 0$, i.e. $\alpha_i^0 \mathrm{parity}(x_i^0) > 0$ and $\alpha_i^1 \mathrm{parity}(x_i^1) > 0$. Since this is true for all $x_i^0$ in $H_d^0$, we have proved lemma 2.3. $\square$

**Theorem 2.4.** *Let $f(x) = b + \sum_{i=1}^{2^d} \alpha_i K_\sigma(x_i, x)$ be an affine combination of Gaussians with same width $\sigma$ centered on points $x_i \in X_d$. If $f$ solves the parity problem, then there are at least $2^{d-1}$ non-zero coefficients $\alpha_i$.*

*Proof.* We begin with two preliminary results. First, given any $x_i \in X_d$, the number of points in $X_d$ that differ from $x_i$ by exactly $k$ bits is $\binom{d}{k}$. Thus,

$$\sum_{x_j \in X_d} K_\sigma(x_i, x_j) = \sum_{k=0}^{d} \binom{d}{k} \exp\left(-\frac{k^2}{2\sigma^2}\right) = c_\sigma. \tag{7}$$

Second, it is possible to find a linear combination (i.e. without bias) of Gaussians $g$ such that $g(x_i) = f(x_i)$ for all $x_i \in X_d$. Indeed, let

$$g(x) = f(x) - b + \sum_{x_j \in X_d} \beta_j K_\sigma(x_j, x). \tag{8}$$

$g$ verifies $g(x_i) = f(x_i)$ iff $\sum_{x_j \in X_d} \beta_j K_\sigma(x_j, x_i) = b$, i.e. the vector $\beta$ satisfies the linear system $M_\sigma \beta = b\mathbf{1}$, where $M_\sigma$ is the kernel matrix whose element $(i,j)$ is $K_\sigma(x_i, x_j)$ and $\mathbf{1}$ is a vector of ones. It is well known that $M_\sigma$ is invertible as long as the $x_i$ are all different, which is the case here (Micchelli, 1986). Thus $\beta = bM_\sigma^{-1}\mathbf{1}$ is the only solution to the system.
We now proceed to the proof of the theorem. By contradiction, suppose $f$ solves the parity problem with less than $2^{d-1}$ non-zero coefficients $\alpha_i$. Then there exist two points $x_s$ and $x_t$ in $X_d$ such that $\alpha_s = \alpha_t = 0$ and $\mathrm{parity}(x_s) = 1 = -\mathrm{parity}(x_t)$. Consider the function $g$ defined as in eq. 8 with $\beta = bM_\sigma^{-1}\mathbf{1}$. Since $g(x_i) = f(x_i)$ for all $x_i \in X_d$, $g$ solves the parity problem with a linear combination of Gaussians centered points in $X_d$. Thus, applying lemma 2.3, we have in particular that $\beta_s \mathrm{parity}(x_s) > 0$ and $\beta_t \mathrm{parity}(x_t) > 0$ (because $\alpha_s = \alpha_t = 0$), so that $\beta_s \beta_t < 0$. But, because of eq. 7, $M_\sigma \mathbf{1} = c_\sigma \mathbf{1}$, which means $\mathbf{1}$ is an eigenvector of $M_\sigma$ with eigenvalue $c_\sigma > 0$. Consequently, $\mathbf{1}$ is also an eigenvector of $M_\sigma^{-1}$ with eigenvalue $c_\sigma^{-1} > 0$, and $\beta = bM_\sigma^{-1}\mathbf{1} = bc_\sigma^{-1}\mathbf{1}$, which is in contradiction with $\beta_s \beta_t < 0$: $f$ must therefore have at least $2^{d-1}$ non-zero coefficients. $\square$

The bound in theorem 2.4 is tight, since it is possible to solve the parity problem with exactly $2^{d-1}$ Gaussians and a bias, for instance by using a negative bias and putting a positive weight on each example satisfying $\mathrm{parity}(x_i) = 1$. When trained to learn the parity function, a SVM may learn a function that looks like the opposite of the parity on test points (while still performing optimally on training points), but it is an artefact of the specific geometry of the problem, and only occurs when the training set size is appropriate compared to $|X_d| = 2^d$ (see (Bengio, Delalleau and Le Roux, 2005) for details). Note that if the centers of the Gaussians are not restricted anymore to be points in $X_d$, it is possible to solve the parity problem with only $d + 1$ Gaussians and no bias (Bengio, Delalleau and Le Roux, 2005).

One may argue that parity is a simple discrete toy problem of little interest. But even if we have to restrict the analysis to discrete samples in $\{0,1\}^d$ for mathematical reasons, the parity function can be extended to a smooth function on the $[0,1]^d$ hypercube depending only on the continuous sum $b_1 + \ldots + b_d$. Theorem 2.4 is thus a basis to argue that the number of Gaussians needed to learn a function with many variations in a continuous space may scale linearly with these variations, and thus possibly exponentially in the dimension.

## 2.2 Results for Semi-Supervised Learning

In this section we focus on algorithms of the type described in recent papers (Zhu, Ghahramani and Lafferty, 2003; Zhou et al., 2004; Belkin, Matveeva and Niyogi, 2004; Delalleau, Bengio and Le Roux, 2005), which are graph-based non-parametric semi-supervised learning algorithms. Note that transductive SVMs, which are another class of semi-supervised algorithms, are already subject to the limitations of corollary 2.2. The graph-based algorithms we consider here can be seen as minimizing the following cost function, as shown in (Delalleau, Bengio and Le Roux, 2005):

$$C(\hat{Y}) = \|\hat{Y}_l - Y_l\|^2 + \mu \hat{Y}^\top L \hat{Y} + \mu \epsilon \|\hat{Y}\|^2 \tag{9}$$

with $\hat{Y} = (\hat{y}_1, \ldots, \hat{y}_n)$ the estimated labels on both labeled and unlabeled data, and $L$ the (un-normalized) graph Laplacian derived from a similarity function $W$ between points such that $W_{ij} = W(x_i, x_j)$ corresponds to the weights of the edges in the graph. Here, $\hat{Y}_l = (\hat{y}_1, \ldots, \hat{y}_l)$ is the vector of estimated labels on the $l$ labeled examples, whose known labels are given by $Y_l = (y_1, \ldots, y_l)$, and one may constrain $\hat{Y}_l = Y_l$ as in (Zhu, Ghahramani and Lafferty, 2003) by letting $\mu \to 0$. We define a region with constant label as a connected subset of the graph where all nodes $x_i$ have the same estimated label (sign of $\hat{y}_i$), and such that no other node can be added while keeping these properties.

**Proposition 2.5.** *After running a label propagation algorithm minimizing the cost of eq. 9, the number of regions with constant estimated label is less than (or equal to) the number of labeled examples.*

*Proof.* By contradiction, if this proposition is false, then there exists a region with constant estimated label that does not contain any labeled example. Without loss of generality, consider the case of a positive constant label, with $x_{l+1}, \ldots, x_{l+q}$ the $q$ samples in this region. The part of the cost of eq. 9 depending on their labels is

$$C(\hat{y}_{l+1}, \ldots, \hat{y}_{l+q}) = \frac{\mu}{2} \sum_{i,j=l+1}^{l+q} W_{ij}(\hat{y}_i - \hat{y}_j)^2$$

$$+ \mu \sum_{i=l+1}^{l+q} \left( \sum_{j \notin \{l+1, \ldots, l+q\}} W_{ij}(\hat{y}_i - \hat{y}_j)^2 \right) + \mu \epsilon \sum_{i=l+1}^{l+q} \hat{y}_i^2.$$

The second term is stricly positive, and because the region we consider is maximal (by definition) all samples $x_j$ outside of the region such that $W_{ij} > 0$ verify $\hat{y}_j < 0$ (for $x_i$ a sample in the region). Since all $\hat{y}_i$ are strictly positive for $i \in \{l+1, \ldots, l+q\}$, this means this second term can be stricly decreased by setting all $\hat{y}_i$ to 0 for $i \in \{l+1, \ldots, l+q\}$. This also sets the first and third terms to zero (i.e. their minimum), showing that the set of labels $\hat{y}_i$ are not optimal, which conflicts with their definition as labels minimizing $C$.  □

This means that if the class distributions are such that there are many distinct regions with constant labels (either separated by low-density regions or regions with samples from the other class), we will need at least the same number of labeled samples as there are such regions (assuming we are using a sparse local kernel such as the $k$-nearest neighbor kernel, or a thresholded Gaussian kernel). But this number could *grow exponentially with the dimension of the manifold(s) on which the data lie*, for instance in the case of a labeling function varying highly along each dimension, *even if the label variations are "simple" in a non-local sense*, e.g. if they alternate in a regular fashion. When the kernel is not sparse (e.g. Gaussian kernel), obtaining such a result is less obvious. However, there often exists a sparse approximation of the kernel. Thus we conjecture the same kind of result holds for dense weight matrices, if the weighting function is local in the sense that it is close to zero when applied to a pair of examples far from each other.

## 3 Extensions and Conclusions

In (Bengio, Delalleau and Le Roux, 2005) we present additional results that apply to unsupervised learning algorithms such as non-parametric manifold learning algorithms (Roweis and Saul, 2000; Tenenbaum, de Silva and Langford, 2000; Schölkopf, Smola and Müller, 1998; Belkin and Niyogi, 2003). We find that when the underlying manifold varies a lot in the sense of having high curvature in many places, then a large number of examples is required. Note that the tangent plane is defined by the derivatives of the kernel machine function $f$, for such algorithms. The core result is that the manifold tangent plane at $x$ is mostly defined by the near neighbors of $x$ in the training set (more precisely it is constrained to be in the span of the vectors $x - x_i$, with $x_i$ a neighbor of $x$). Hence one needs to cover the manifold with small enough linear patches with at least $d + 1$ examples per patch (where $d$ is the dimension of the manifold).

In the same paper, we present a conjecture that generalizes the results presented here for Gaussian kernel classifiers to a larger class of local kernels, using the same notion of locality of the derivative summarized above for manifold learning algorithms. In that case the derivative of $f$ represents the normal of the decision surface, and we find that at $x$ it mostly depends on the neighbors of $x$ in the training set.

It could be argued that if a function has many local variations (hence is not very smooth), then it is not learnable unless having strong prior knowledge at hand. However, this is not true. For example consider functions that have low Kolmogorov complexity, i.e. can be described by a short string in some language. The only prior we need in order to quickly learn such functions (in terms of number of examples needed) is that functions that are simple to express in that language (e.g. a programming language) are preferred. For example, the functions $g(x) = sin(x)$ or $g(x) = parity(x)$ would be easy to learn using the C programming language to define the prior, even though the number of variations of $g(x)$ can be chosen to be arbitrarily large (hence also the number of required training examples when using only the smoothness prior), while keeping the Kolmogorov complexity constant. We do not propose to necessarily focus on the Kolmogorov complexity to design new learning algorithms, but we use this example to illustrate that it is possible to learn apparently complex functions (because they vary a lot), as long as one uses a "non-local" learning algorithm, corresponding to a broad prior, not solely relying on the smoothness prior. Of course, if additional domain knowledge about the task is available, it should be used, but without abandoning research on learning algorithms that can address a wider scope of problems. We hope that this paper will stimulate more research into such learning algorithms, since we expect local learning algorithms (that only rely on the smoothness prior) will be insufficient to make significant progress on complex problems such as those raised by research on Artificial Intelligence.

**Acknowledgments**

The authors would like to thank the following funding organizations for support: NSERC, MITACS, and the Canada Research Chairs. The authors are also grateful for the feedback and stimulating exchanges that helped shape this paper, with Yann Le Cun and Léon Bottou, as well as for the anonymous reviewers' helpful comments.

## References

Belkin, M., Matveeva, I., and Niyogi, P. (2004). Regularization and semi-supervised learning on large graphs. In Shawe-Taylor, J. and Singer, Y., editors, *COLT'2004*. Springer.

Belkin, M. and Niyogi, P. (2003). Using manifold structure for partially labeled classification. In Becker, S., Thrun, S., and Obermayer, K., editors, *Advances in Neural*

*Information Processing Systems 15*, Cambridge, MA. MIT Press.

Bengio, Y., Delalleau, O., and Le Roux, N. (2005). The curse of dimensionality for local kernel machines. Technical Report 1258, Département d'informatique et recherche opérationnelle, Université de Montréal.

Bengio, Y., Delalleau, O., Le Roux, N., Paiement, J.-F., Vincent, P., and Ouimet, M. (2004). Learning eigenfunctions links spectral embedding and kernel PCA. *Neural Computation*, 16(10):2197–2219.

Boser, B., Guyon, I., and Vapnik, V. (1992). A training algorithm for optimal margin classifiers. In *Fifth Annual Workshop on Computational Learning Theory*, pages 144–152, Pittsburgh.

Brand, M. (2003). Charting a manifold. In Becker, S., Thrun, S., and Obermayer, K., editors, *Advances in Neural Information Processing Systems 15*. MIT Press.

Cortes, C. and Vapnik, V. (1995). Support vector networks. *Machine Learning*, 20:273–297.

Delalleau, O., Bengio, Y., and Le Roux, N. (2005). Efficient non-parametric function induction in semi-supervised learning. In Cowell, R. and Ghahramani, Z., editors, *Proceedings of the Tenth International Workshop on Artificial Intelligence and Statistics, Jan 6-8, 2005, Savannah Hotel, Barbados*, pages 96–103. Society for Artificial Intelligence and Statistics.

Duda, R. and Hart, P. (1973). *Pattern Classification and Scene Analysis*. Wiley, New York.

Härdle, W., Müller, M., Sperlich, S., and Werwatz, A. (2004). *Nonparametric and Semiparametric Models*. Springer, http://www.xplore-stat.de/ebooks/ebooks.html.

Micchelli, C. A. (1986). Interpolation of scattered data: distance matrices and conditionally positive definite functions. *Constructive Approximation*, 2:11–22.

Roweis, S. and Saul, L. (2000). Nonlinear dimensionality reduction by locally linear embedding. *Science*, 290(5500):2323–2326.

Schmitt, M. (2002). Descartes' rule of signs for radial basis function neural networks. *Neural Computation*, 14(12):2997–3011.

Schölkopf, B., Burges, C. J. C., and Smola, A. J. (1999). *Advances in Kernel Methods — Support Vector Learning*. MIT Press, Cambridge, MA.

Schölkopf, B., Smola, A., and Müller, K.-R. (1998). Nonlinear component analysis as a kernel eigenvalue problem. *Neural Computation*, 10:1299–1319.

Snapp, R. R. and Venkatesh, S. S. (1998). Asymptotic derivation of the finite-sample risk of the k nearest neighbor classifier. Technical Report UVM-CS-1998-0101, Department of Computer Science, University of Vermont.

Tenenbaum, J., de Silva, V., and Langford, J. (2000). A global geometric framework for nonlinear dimensionality reduction. *Science*, 290(5500):2319–2323.

Weiss, Y. (1999). Segmentation using eigenvectors: a unifying view. In *Proceedings IEEE International Conference on Computer Vision*, pages 975–982.

Williams, C. and Rasmussen, C. (1996). Gaussian processes for regression. In Touretzky, D., Mozer, M., and Hasselmo, M., editors, *Advances in Neural Information Processing Systems 8*, pages 514–520. MIT Press, Cambridge, MA.

Zhou, D., Bousquet, O., Navin Lal, T., Weston, J., and Schölkopf, B. (2004). Learning with local and global consistency. In Thrun, S., Saul, L., and Schölkopf, B., editors, *Advances in Neural Information Processing Systems 16*, Cambridge, MA. MIT Press.

Zhu, X., Ghahramani, Z., and Lafferty, J. (2003). Semi-supervised learning using Gaussian fields and harmonic functions. In *ICML'2003*.
